# Learning Topology with the Generative Gaussian Graph and the EM Algorithm

**Michaël Aupetit**
CEA - DASE
BP 12 - 91680
Bruyères-le-Châtel, France
`aupetit@dase.bruyeres.cea.fr`

## Abstract

Given a set of points and a set of prototypes representing them, how to create a graph of the prototypes whose topology accounts for that of the points? This problem had not yet been explored in the framework of statistical learning theory. In this work, we propose a generative model based on the Delaunay graph of the prototypes and the Expectation-Maximization algorithm to learn the parameters. This work is a first step towards the construction of a topological model of a set of points grounded on statistics.

## 1 Introduction

### 1.1 Topology what for?

Given a set of points in a high-dimensional euclidean space, we intend to extract the topology of the manifolds from which they are drawn. There are several reasons for this among which: increasing our knowledge about this set of points by measuring its topological features (connectedness, intrinsic dimension, Betti numbers (number of voids, holes, tunnels...)) in the context of exploratory data analysis [1], allowing to compare two sets of points *wrt* their topological characteristics or to find clusters as connected components in the context of pattern recognition [2], or finding shortest path along manifolds in the context of robotics [3].

There are two families of approaches which deal with "topology" : on one hand, the "topology preserving" approaches based on nonlinear projection of the data in lower dimensional spaces with a constrained topology to allow visualization [4, 5, 6, 7, 8]; on the other hand, the "topology modelling" approaches based on the construction of a structure whose topology is not constrained *a priori*, so it is expected to better account for that of the data [9, 10, 11] at the expense of the visualisability. Much work has been done about the former problem also called "manifold learning", from Generative Topographic Mapping [4] to Multi-Dimensional Scaling and its variants [5, 6], Principal Curves [7] and so on. In all these approaches, the intrinsic dimension of the model is fixed *a priori* which eases the visualization but arbitrarily forces the topology of the model. And when the dimension is not fixed as in the mixture of Principal Component Analyzers [8], the connectedness is lost. The latter problem we deal with had never been explored in the statistical learning

perspective. Its aim is not to project and visualize a high-dimensional set of points, but to extract the topological information from it directly in the high-dimensional space, so that the model must be freed as much as possible from any *a priori* topological constraint.

## 1.2 Learning topology: a state of the art

As we may learn a complicated function combining simple basis functions, we shall learn a complicated manifold[1] combining simple basis manifolds. A simplicial complex[2] is such a model based on the combination of simplices, each with its own dimension (a 1-simplex is a line segment, a 2-simplex is a triangle...a $k$-simplex is the convex hull of a set of $k+1$ points). In a simplicial complex, the simplices are exclusively connected by their vertices or their faces. Such a structure is appealing because it is possible to extract from it topological information like Betti numbers, connectedness and intrinsic dimension [10]. A particular simplicial complex is the Delaunay complex defined as the set of simplices whose Voronoï cells[3] of the vertices are adjacent assuming general position for the vertices. The Delaunay graph is made of vertices and edges of the Delaunay complex [12].

All the previous work about topology modelling is grounded on the result of Edelsbrunner and Shah [13] which prove that given a manifold $\mathcal{M} \subset \mathbb{R}^D$ and a set of $N_0$ vector prototypes $\underline{w} \in (\mathbb{R}^D)^{N_0}$ nearby $\mathcal{M}$, it exists a simplicial subcomplex of the Delaunay complex of $\underline{w}$ which has the same topology as $\mathcal{M}$ under what we call the "ES-conditions".

In the present work, the manifold $\mathcal{M}$ is not known but through a finite set of $M$ data points $\underline{v} \in \mathcal{M}^M$. Martinetz and Schulten proposed to build a graph of the prototypes with an algorithm called "Competitive Hebbian Learning" (CHL)[11] to tackle this problem. Their approach has been extended to simplicial complexes by De Silva and Carlsson with the definition of "weak witnesses" [10]. In both cases, the ES-conditions about $\mathcal{M}$ are weakened so they can be verified by a finite sample $\underline{v}$ of $\mathcal{M}$, so that the graph or the simplicial complex built over $\underline{w}$ is proved to have the same topology as $\mathcal{M}$ if $\underline{v}$ is a sufficiently dense sampling of $\mathcal{M}$.

The CHL consists in connecting two prototypes in $\underline{w}$ if they are the first and the second closest neighbors to a point of $\underline{v}$ (closeness *wrt* the Euclidean norm). Each point of $\underline{v}$ leads to an edge, and is called a "weak witness" of the connected prototypes [10]. The topology representing graph obtained is a subgraph of the Delaunay graph. The region of $\mathbb{R}^D$ in which any data point would connect the same prototypes, is the "region of influence" (ROI) of this edge (see Figure 2 d-f). This principle is extended to create $k$-simplices connecting $k+1$ prototypes, which are part of the Delaunay simplicial-complex of $\underline{w}$ [10].

Therefore, the model obtained is based on regions of influence: a simplex exists in the model if there is at least one datum in its ROI. Hence, the capacity of this model to correctly represent the topology of a set of points, strongly depends on the shape and location of the ROI *wrt* the points, and on the presence of noise in the data. Moreover, as far as $N_0 > 2$, it cannot exist an isolated prototype allowing to represent an isolated bump in the data distribution, because any datum of this bump will have two closest prototypes to connect to each other. An aging process has been proposed by Martinetz and Schulten to filter out the noise, which works roughly such that edges with fewer data than a threshold in there ROI are pruned from the graph. This looks like a filter based on the probability density of the data distribution, but no statistical criterion is proposed to tune the parameters. Moreover the area of the ROI may be intractable in high dimension and is not trivially related to the

corresponding line segment, so measuring the frequency over such a region is not relevant to define a useful probability density. At last, the line segment associated to an edge of the graph is not part of the model: data are not projected on it, data drawn from such a line segment may not give rise to the corresponding edge, and the line segment may not intersect at all its associated ROI. In other words, the model is not self-consistent, that is the geometrical realization of the graph is not always a good model of its own topology whatever the density of the sampling.

We proposed to define Voronoï cells of line segments as ROI for the edges and defined a criterion to cut edges with a lower density of data projecting on their middle than on their borders [9]. This solves some of the CHL limits but it still remains one important problem common to both approaches: they rely on the visual control of their quality, *i.e.* no criterion allows to assess the quality of the model especially in dimension greater than 3.

### 1.3   Emerging topology from a statistical generative model

For all the above reasons, we propose another way for modelling topology. The idea is to construct a "good" statistical generative model of the data taking the noise into account, and to assume that its topology is therefore a "good" model of the topology of the manifold which generated the data. The only constraint we impose on this generative model is that its topology must be as "flexible" as possible and must be "extractible". "Flexible" to avoid at best any *a priori* constraint on the topology so as to allow the modelling of any one. "Extractible" to get a "white box" model from which the topological characteristics are tractable in terms of computation. So we propose to define a "generative simplicial complex". However, this work being preliminary, we expose here the simpler case of defining a "generative graph" (a simplicial complex made only of vertices and edges) and tuning its parameters. This allows to demonstrate the feasibility of this approach and to foresee future difficulties when it is extended to simplicial complexes.

It works as follows. Given a set of prototypes located over the data distribution using *e.g.* Vector Quantization [14], the Delaunay graph (DG) of the prototypes is constructed [15]. Then, each edge and each vertex of the graph is the basis of a generative model so that the graph generates a mixture of gaussian density functions. The maximization of the likelihood of the data *wrt* the model, using Expectation-Maximization, allows to tune the weights of this mixture and leads to the emergence of the expected topology representing graph through the edges with non-negligible weights that remain after the optimization process.

We first present the framework and the algorithm we use in section 2. Then we test it on artificial data in section 3 before the discussion and conclusion in section 4.

## 2   A Generative Gaussian Graph to learn topology

### 2.1   The Generative Gaussian Graph

In this work, $\mathcal{M}$ is the support of the probability density function (pdf) $p$ from which are drawn the data $\underline{v}$. In fact, this is not the topology of $\mathcal{M}$ which is of interest, but the topology of manifolds $\mathcal{M}^{prin}$ called "principal manifolds" of the distribution $p$ (in reference to the definition of Tibshirani [7]) which can be viewed as the manifold $\mathcal{M}$ without the noise. We assume the data have been generated by some set of points and segments constituting the set of manifolds $\mathcal{M}^{prin}$ which have been corrupted with additive spherical gaussian noise with mean 0 and unknown variance $\sigma_{noise}^2$. Then, we define a gaussian mixture model to account for the observed data, which is based on both gaussian kernels that we call "gaussian-points", and what we call "gaussian-segments", forming a "Generative Gaussian Graph" (GGG).

The value at point $v_j \in \underline{v}$ of a normalized gaussian-point centered on a prototype $w_i \in \underline{w}$ with variance $\sigma^2$ is defined as: $g^0(v_j, w_i, \sigma) = (2\pi\sigma^2)^{-D/2} \exp(-\frac{(v_j - w_i)^2}{2\sigma^2})$

A normalized gaussian-segment is defined as the sum of an infinite number of gaussian-points evenly spread on a line segment. Thus, this is the integral of a gaussian-point along a line segment. The value at point $v_j$ of the gaussian-segment $[w_{a_i} w_{b_i}]$ associated to the $i^{th}$ edge $\{a_i, b_i\}$ in DG with variance $\sigma^2$ is:

$$
\begin{aligned}
g^1(v_j, \{w_{a_i}, w_{b_i}\}, \sigma) &= \frac{\int_{w_{a_i}}^{w_{b_i}} \exp\left(-\frac{(v_j - w)^2}{2\sigma^2}\right) dw}{(2\pi\sigma^2)^{\frac{D}{2}} L_{a_i b_i}} \\
&= \frac{\exp\left(-\frac{(v_j - q_i^j)^2}{2\sigma^2}\right)}{(2\pi\sigma^2)^{\frac{D-1}{2}}} \cdot \frac{\text{erf}\left(\frac{Q_{a_i b_i}^j}{\sigma\sqrt{2}}\right) - \text{erf}\left(\frac{Q_{a_i b_i}^j - L_{a_i b_i}}{\sigma\sqrt{2}}\right)}{2L_{a_i b_i}}
\end{aligned}
\tag{1}
$$

where $L_{a_i b_i} = \|w_{b_i} - w_{a_i}\|$, $Q_{a_i b_i}^j = \frac{\langle v_j - w_{a_i} | w_{b_i} - w_{a_i}\rangle}{L_{a_i b_i}}$ and $q_i^j = w_{a_i} + (w_{b_i} - w_{a_i})\frac{Q_{a_i b_i}^j}{L_{a_i b_i}}$ is the orthogonal projection of $v_j$ on the straight line passing through $w_{a_i}$ and $w_{b_i}$. In the case where $w_{a_i} = w_{b_i}$, we set $g^1(v_j, \{w_{a_i}, w_{b_i}\}, \sigma) = g^0(v_j, w_{a_i}, \sigma)$.

The left part of the dot product accounts for the gaussian noise orthogonal to the line segment, and the right part for the gaussian noise integrated along the line segment. The functions $g^0$ and $g^1$ are positive and we can prove that: $\int_{\mathbb{R}^D} g^0(v, w_i, \sigma)dv = 1$ and $\int_{\mathbb{R}^D} g^1(v, \{w_a, w_b\}, \sigma)dv = 1$, so they are both probability density functions. A gaussian-point is associated to each prototype in $\underline{w}$ and a gaussian-segment to each edge in DG.

The gaussian mixture is obtained by a weighting sum of the $N_0$ gaussian-points and $N_1$ gaussian-segments, such that the weights $\underline{\pi}$ sum to 1 and are non-negative:

$$
p(v_j | \underline{\pi}, \underline{w}, \sigma, DG) = \sum_{k=0}^{1} \sum_{i=1}^{N_k} \pi_i^k g^k(v_j, s_i^k, \sigma)
\tag{2}
$$

with $\sum_{k=0}^{1} \sum_{i=1}^{N_k} \pi_i^k = 1$ and $\forall i, k, \; \pi_i^k \geq 0$, where $s_i^0 = w_i$ and $s_i^1 = \{w_{a_i}, w_{b_i}\}$ such that $\{a_i, b_i\}$ is the $i^{th}$ edge in DG. The weight $\pi_i^0$ (resp. $\pi_i^1$) is the probability that a datum $v$ was drawn from the gaussian-point associated to $w_i$ (resp. the gaussian-segment associated to the $i^{th}$ edge of $DG$).

## 2.2 Measure of quality

The function $p(v_j | \underline{\pi}, \underline{w}, \sigma, DG)$ is the probability density at $v_j$ given the parameters of the model. We measure the likelihood $P$ of the data $\underline{v}$ *wrt* the parameters of the GGG model:

$$
P = P(\underline{\pi}, \underline{w}, \sigma, DG) = \prod_{j=1}^{M} p(v_j | \underline{\pi}, \underline{w}, \sigma, DG)
\tag{3}
$$

## 2.3 The Expectation-Maximization algorithm

In order to maximize the likelihood $P$ or equivalently to minimize the negative log-likelihood $L = -\log(P)$ *wrt* $\underline{\pi}$ and $\sigma$, we use the Expectation-Maximization algorithm.

We refer to [2] (pages $59 - 73$) and [16] for further details. The minimization of the negative log-likelihood consists in $t_{max}$ iterative steps updating $\underline{\pi}$ and $\sigma$ which ensure the decrease of $L$. The updating rules take into account the constraints about positivity or sum to unity of the parameters:

$$
\begin{aligned}
\pi_i^{k[\text{new}]} &= \tfrac{1}{M} \sum_{j=1}^{M} P(k,i|v_j) \\
\sigma^{2[\text{new}]} &= \tfrac{1}{DM} \sum_{j=1}^{M} [\sum_{i=1}^{N_0} P(0,i|v_j)(v_j - w_i)^2 \\
&\quad + \sum_{i=1}^{N_1} P(1,i|v_j) \frac{(2\pi\sigma^2)^{-D/2} \exp(-\frac{(v_j - q_i^j)^2}{2\sigma^2})(I_1[(v_j - q_i^j)^2 + \sigma^2] + I_2)}{L_{a_i b_i} \cdot g^1(v_j, \{w_{a_i}, w_{b_i}\}, \sigma)}]
\end{aligned}
\tag{4}
$$

where

$$
\begin{aligned}
I_1 &= \sigma \sqrt{\tfrac{\pi}{2}} (\text{erf}(\tfrac{Q_{a_i b_i}^j}{\sigma\sqrt{2}}) - \text{erf}(\tfrac{Q_{a_i b_i}^j - L_{a_i b_i}}{\sigma\sqrt{2}})) \\
I_2 &= \sigma^2 \left( (Q_{a_i b_i}^j - L_{a_i b_i}) \exp(-\tfrac{(Q_{a_i b_i}^j - L_{a_i b_i})^2}{2\sigma^2}) - Q_{a_i b_i}^j \exp(-\tfrac{(Q_{a_i b_i}^j)^2}{2\sigma^2}) \right)
\end{aligned}
\tag{5}
$$

and $P(k,i|v_j) = \frac{\pi_i^k g^k(v_j, s_i^k, \sigma)}{p(v_j|\underline{\pi},\underline{w},\sigma,DG)}$ is the posterior probability that the datum $v_j$ was generated by the component associated to $(k, i)$.

### 2.4 Emerging topology by maximizing the likelihood

Finally, to get the topology representing graph from the generative model, the core idea is to prune from the initial $DG$ the edges for which there is probability $\epsilon$ they generated the data. The complete algorithm is the following:

1. Initialize the location of the prototypes $\underline{w}$ using vector quantization [14].
2. Construct the Delaunay graph DG of the prototypes.
3. Initialize the weights $\underline{\pi}$ to $1/(N_0 + N_1)$ to give equiprobability to each vertices and edges.
4. Given $\underline{w}$ and DG, use updating rules (4) to find $\sigma^{2*}$ and $\underline{\pi}^*$ maximizing the likelihood $P$.
5. Prune the edges $\{a_i b_i\}$ of DG associated to the gaussian segments with probability $\pi_i^1 \leq \epsilon$ where $\pi_i^1 \in \underline{\pi}^*$.

The topology representing graph emerges from the edges with probabilities $\underline{\pi}^* > \epsilon$. It is the graph which best models the topology of the data in the sense of the maximum likelihood *wrt* $\underline{\pi}$, $\sigma$, $\epsilon$ and the set of prototypes $\underline{w}$ and their Delaunay graph.

## 3 Experiments

In these experiments, given a set of points and a set of prototypes located thanks to vector quantization [14], we want to verify the relevance of the GGG to learn the topology in various noise conditions. The principle of the GGG is shown in the Figure 1. In the Figure 2, we show the comparison of the GGG to a CHL for which we filter out edges which have a number of hits lower than a threshold $T$. The data and prototypes are the same for both algorithms. We set $T^*$ such that the graph obtained matches visually as close as possible the expected solution. We optimize $\sigma$ and $\underline{\pi}$ using (4) for $t_{max} = 100$ steps and $\epsilon = 0.001$. Conditions and conclusions of the experiments are given in the captions.

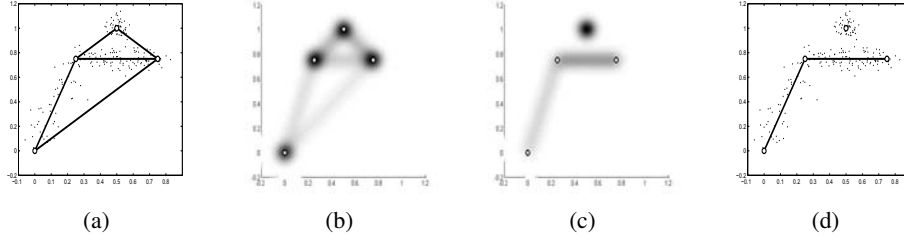

|     (a)     |     (b)     |     (c)     |     (d)     |

Figure 1: **Principle of the Generative Gaussian Graph**: (a) Data drawn from an oblique segment, an horizontal one and an isolated point with respective density $\{0.25; 0.5; 0.25\}$. The prototypes are located at the extreme points of the segments, and at the isolated point. They are connected with edges from the Delaunay graph. (b) The corresponding initial Generative Gaussian Graph. (c) The optimal GGG obtained after optimization of the likelihood according to $\sigma$ and $\underline{\pi}$. (d) The edges of the optimal GGG associated to non-negligible probabilities model the topology of the data.

## 4    Discussion

We propose that the problem of learning the topology of a set of points can be posed as a statistical learning problem: we assume that the topology of a statistical generative model of a set of points is an estimator of the topology of the principal manifold of this set. From this assumption, we define a topologically flexible statistical generative mixture model that we call Generative Gaussian Graph from which we can extract the topology. The final topology representing graph emerges from the edges with non-negligible probability. We propose to use the Delaunay graph as an initial graph assuming it is rich enough to contain as a subgraph a good topological model of the data. The use of the likelihood criterion makes possible cross-validation to select the best generative model hence the best topological model in terms of generalization capacities.

The GGG allows to avoid the limits of the CHL for modelling topology. In particular, it allows to take into account the noise and to model isolated bumps. Moreover, the likelihood of the data *wrt* the GGG is maximized during the learning, allowing to measure the quality of the model even when no visualization is possible. For some particular data distributions where all the data lie on the Delaunay line segments, no maximum of the likelihood exists. This case is not a problem because $\sigma = 0$ effectively defines a good solution (no noise in a data set drawn from a graph). If only some of the data lie exactly on the line segments, a maximum of the likelihood still exists because $\sigma^2$ defines the variance for all the generative gaussian points and segments at the same time so it cannot vanish to $0$. The computing time complexity of the GGG is $o(D(N_0 + N_1)Mt_{max})$ plus the time $O(DN_0^3)$ [15] needed to build the Delaunay graph which dominates the overall worst time complexity. The Competitive Hebbian Learning is in time $o(DN_0M)$. As in general, the CHL builds too much edges than needed to model the topology, it would be interesting to use the Delaunay subgraph obtained with the CHL as a starting point for the GGG model.

The Generative Gaussian Graph can be viewed as a generalization of gaussian mixtures to points and segments: a gaussian mixture is a GGG with no edge. GGG provides at the same time an estimation of the data distribution density more accurate than the gaussian mixture based on the same set of prototypes and the same noise isovariance hypothesis (because it adds gaussian-segments to the pool of gaussian-points), and intrinsically an explicit model of the topology of the data set which provides most of the topological information at once. In contrast, other generative models do not provide any insight about the topology of the data, except the Generative Topographic Map (GTM) [4], the revisited Principal Manifolds [7] or the mixture of Probabilistics Principal Component Analysers (PPCA) [8]. However, in the two former cases, the intrinsic dimension of the model is fixed *a priori* and

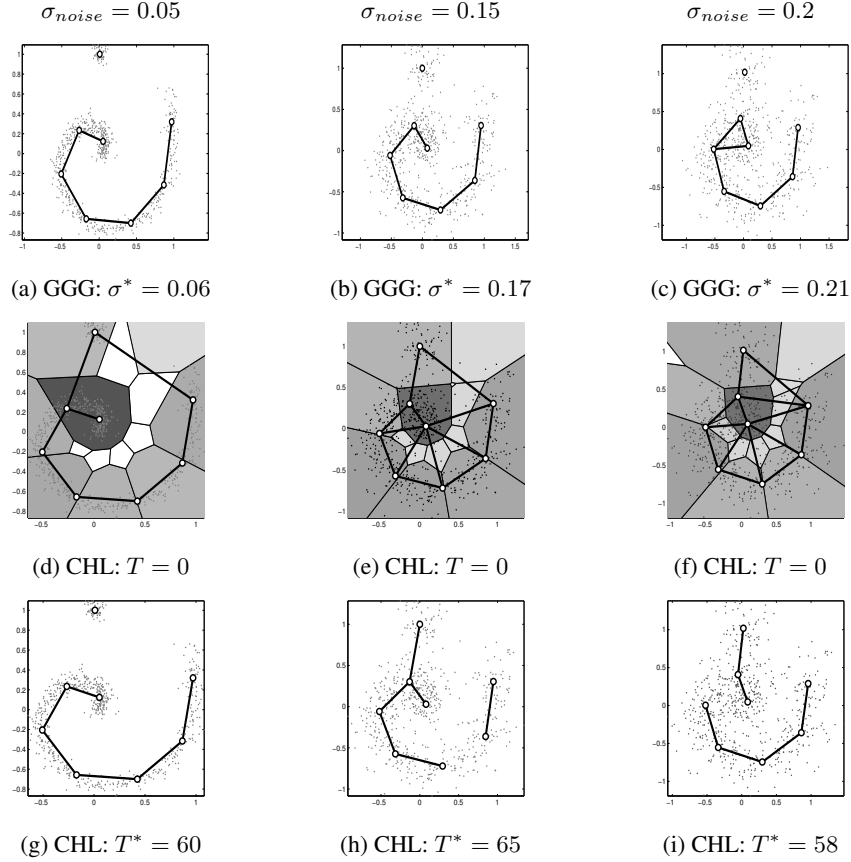

Figure 2: **Learning the topology of a data set**: 600 data drawn from a spirale and an isolated point corrupted with additive gaussian noise with mean 0 and variance $\sigma_{noise}^2$. Prototypes are located by vector quantization [14]. (a-c) The edges of the GGG with weights greater than $\epsilon$ allow to recover the topology of the principal manifolds except for large noise variance (c) where a triangle was created at the center of the spirale. $\sigma^*$ over-estimates $\sigma_{noise}$ because the model is piecewise linear while the true manifolds are non-linear. (d-f) The CHL without threshold (T=0) is not able to recover the true topology of the data for even small $\sigma_{noise}$. In particular, the isolated bump cannot be recovered. The grey cells correspond to ROI of the edges (darker cells contain more data). It shows these cells are not intuitively related to the edges they are associated to (*e.g.* they may have very tiny areas (e), and may partly (d) or never (f) contain the corresponding line segment). (g-h) The CHL with a threshold $T$ allows to recover the topology of the data only for small noise variance (g) (Notice $T_1 < T_2 \Rightarrow DG_{CHL}(T_2) \subseteq DG_{CHL}(T_1)$). Moreover, setting $T$ requires visual control and is not associated to the optimum of any energy function which prevents its use in higher dimensional space.

not learned from the data, while in the latter the local intrinsic dimension is learned but the connectedness between the local models is not.

One obvious way to follow to extend this work is considering a simplicial complex in place of the graph to get the full topological information extractible. Some other interesting questions arise about the curse of the dimension, the selection of the number of prototypes and the threshold $\epsilon$, the theoretical grounding of the connection between the likelihood and some topological measure of accuracy, the possibility to devise a "universal topology estimator", the way to deal with data sets with multi-scale structures or background noise. . .

This preliminary work is an attempt to bridge the gap between Statistical Learning Theory [17] and Computational Topology [18][19]. We wish it to cross-fertilize and to open new perspectives in both fields.

## Footnotes

[1]For simplicity, we call "manifold" what can be actually a set of manifolds connected or not to each other with possibly various intrinsic dimensions.

[2]The terms "simplex" or "graph" denote both the abstract object and its geometrical realization.

[3]Given a set of points $\underline{w}$ in $\mathbb{R}^D$, $V_i = \{v \in \mathbb{R}^D | (v - w_i)^2 \leq (v - w_j)^2, \forall j\}$ defines the Voronoï cell associated to $w_i \in \underline{w}$.

# References

[1] M. Aupetit and T. Catz. High-dimensional labeled data analysis with topology representing graphs. *Neurocomputing, Elsevier*, 63:139–169, 2005.

[2] C. M. Bishop. *Neural Networks for Pattern Recognition*. Oxford Univ. Press, New York, 1995.

[3] M. Zeller, R. Sharma, and K. Schulten. Topology representing network for sensor-based robot motion planning. *World Congress on Neural Networks, INNS Press*, pages 100–103, 1996.

[4] C. M. Bishop, M. Svensén, and C. K. I. Williams. Gtm: the generative topographic mapping. *Neural Computation, MIT Press*, 10(1):215–234, 1998.

[5] V. de Silva and J. B. Tenenbaum. Global versus local methods for nonlinear dimensionality reduction. *In S. Becker, S. Thrun, K. Obermayer (Eds) Advances in Neural Information Processing Systems, MIT Press,Cambridge, MA*, 15:705–712, 2003.

[6] J. A. Lee, A. Lendasse, and M. Verleysen. Curvilinear distance analysis versus isomap. *Europ. Symp. on Art. Neural Networks, Bruges (Belgium), d-side eds.*, pages 185–192, 2002.

[7] R. Tibshirani. Principal curves revisited. *Statistics and Computing*, (2):183–190, 1992.

[8] M. E. Tipping and C. M. Bishop. Mixtures of probabilistic principal component analysers. *Neural Computation*, 11(2):443–482, 1999.

[9] M. Aupetit. Robust topology representing networks. *European Symp. on Artificial Neural Networks, Bruges (Belgium), d-side eds.*, pages 45–50, 2003.

[10] V. de Silva and G. Carlsson. Topological estimation using witness complexes. *In M. Alexa and S. Rusinkiewicz (Eds) Eurographics Symposium on Point-Based Graphics, ETH, Zürich,Switzerland, June 2-4*, 2004.

[11] T. M. Martinetz and K. J. Schulten. Topology representing networks. *Neural Networks, Elsevier London*, 7:507–522, 1994.

[12] A. Okabe, B. Boots, and K. Sugihara. *Spatial tessellations: concepts and applications of Voronoï diagrams*. John Wiley, Chichester, 1992.

[13] H. Edelsbrunner and N. R. Shah. Triangulating topological spaces. *International Journal on Computational Geometry and Applications*, 7:365–378, 1997.

[14] T. M. Martinetz, S. G. Berkovitch, and K. J. Schulten. "neural-gas" network for vector quantization and its application to time-series prediction. *IEEE Trans. on NN*, 4(4):558–569, 1993.

[15] E. Agrell. A method for examining vector quantizer structures. *Proceedings of IEEE International Symposium on Information Theory, San Antonio, TX*, page 394, 1993.

[16] A. Dempster, N. Laird, and D. Rubin. Maximum likelihood from incomplete data via the em algorithm. *Journal of the Royal Statistical Society, Series B*, 39(1):1–38, 1977.

[17] V.N. Vapnik. *Statistical Learning Theory*. John Wiley, 1998.

[18] T. Dey, H. Edelsbrunner, and S. Guha. Computational topology. *In B. Chazelle, J. Goodman and R. Pollack, editors, Advances in Discrete and Computational Geometry. American Math. Society, Princeton, NJ*, 1999.

[19] V. Robins, J. Abernethy, N. Rooney, and E. Bradley. Topology and intelligent data analysis. *IDA-03 (International Symposium on Intelligent Data Analysis), Berlin*, 2003.
